# Geometrical Singularities in the Neuromanifold of Multilayer Perceptrons

**Shun-ichi Amari, Hyeyoung Park, and Tomoko Ozeki**
Brain Science Institute, RIKEN
Hirosawa 2-1, Wako, Saitama, 351-0198, Japan
{*amari, hypark, tomoko*} *@brain.riken.go.jp*

## Abstract

Singularities are ubiquitous in the parameter space of hierarchical models such as multilayer perceptrons. At singularities, the Fisher information matrix degenerates, and the Cramér-Rao paradigm does no more hold, implying that the classical model selection theory such as AIC and MDL cannot be applied. It is important to study the relation between the generalization error and the training error at singularities. The present paper demonstrates a method of analyzing these errors both for the maximum likelihood estimator and the Bayesian predictive distribution in terms of Gaussian random fields, by using simple models.

## 1 Introduction

A neural network is specified by a number of parameters which are synaptic weights and biases. Learning takes place by modifying these parameters from observed input-output examples. Let us denote these parameters by a vector $\boldsymbol{\theta} = (\theta_1, \cdots, \theta_n)$. Then, a network is represented by a point in the parameter space $S$, where $\boldsymbol{\theta}$ plays the role of a coordinate system. The parameter space $S$ is called a neuromanifold.

A learning process is represented by a trajectory in the neuromanifold. The dynamical behavior of learning is known to be very slow, because of the plateau phenomenon. The statistical physical method [1] has made it clear that plateaus are ubiquitous in a large-scale perceptron. In order to improve the dynamics of learning, the natural gradient learning method has been introduced by taking the Riemannian geometrical structure of the neuromanifold into account [2, 3]. Its adaptive version, where the inverse of the Fisher information matrix is estimated adaptively, is shown to have excellent behaviors by computer simulations [4, 5].

Because of the symmetry in the architecture of the multilayer perceptrons, the parameter space of the MLP admits an equivalence relation [6, 7]. The residue class divided by the equivalence relation gives rise to singularities in the neuromanifold, and plateaus exist at such singularities [8]. The Fisher information matrix becomes singular at singularities, so that the neuromanifold is strongly curved like the space-time including black holes.

In the neighborhood of singularities, the Fisher-Cramér-Rao paradigm does not

hold, and the estimator is no more subject to the Gaussian distribution even asymptotically. This is essential in neural learning and model selection. The AIC and MDL criteria of model selection use the Gaussian paradigm, so that it is not appropriate.

The problem was first pointed out by Hagiwara et al. [9]. Watanabe [10] applied algebraic geometry to elucidate the behavior of the Bayesian predictive estimator in MLP, showing sharp difference in regular cases and singular cases. Fukumizu [11] gives a general analysis of the maximum likelihood estimators in singular statistical models including the multilayer perceptrons.

The present paper is a first step to elucidate effects of singularities in the neuromanifold of multilayer perceptrons. We use a simple cone model to elucidate how different the behaviors of the maximum likelihood estimator and the Bayes predictive distribution are from the regular case. To this end, we introduce the Gaussian random field [11, 12, 13], and analyze the generalization error and training error for both the mle (maximum likelihood estimator) and the Bayes estimator.

## 2 Topology of neuromanifold

Let us consider MLP with $h$ hidden units and one output unit,

$$y = \sum_{i=1}^{h} v_i \varphi\left(\boldsymbol{w}_i \cdot \boldsymbol{x}\right) + n. \tag{1}$$

where $y$ is output, $\boldsymbol{x}$ is input and $n$ is Gaussian noise. Let us summarize all the parameters in a single parameter vector $\boldsymbol{\theta} = (\boldsymbol{w}_1, \cdots, \boldsymbol{w}_h; v_1, \cdots, v_h)$ and write

$$f(\boldsymbol{x}; \boldsymbol{\theta}) = \sum_{i=1}^{h} v_i \varphi\left(\boldsymbol{w}_i \cdot \boldsymbol{x}\right). \tag{2}$$

Then, $\boldsymbol{\theta}$ is a coordinate system of the neuromanifold. Because of the noise, the input-output relation is stochastic, given by the conditional probability distribution

$$p(y|\boldsymbol{x}, \boldsymbol{\theta}) = \frac{1}{\sqrt{2}} \exp\left\{-\frac{1}{2}\left(y - f(\boldsymbol{x}; \boldsymbol{\theta})\right)^2\right\}, \tag{3}$$

where we normalized the scale of noise equal to 1. Each point in the neuromanifold represents a neural network or its probability distribution.

It is known that the behavior of MLP is invariant under 1) permutations of hidden units, and 2) sign change of both $\boldsymbol{w}_i$ and $v_i$ at the same time. Two networks are equivalent when they are mapped by any of the above operations which form a group. Hence, it is natural to treat the residual space $S/\approx$, where $\approx$ is the equivalence relation. There are some points which are invariant under a some nontrivial isotropy subgroup, on which singularities occurs.

When $v_i = 0$, $v_i \varphi\left(\boldsymbol{w}_i \cdot \boldsymbol{x}\right) = 0$ so that all the points on the submanifold $v_i = 0$ are equivalent whatever $\boldsymbol{w}_i$ is. We do not need this hidden unit. Hence, in $M = S/\approx$, all of these points are reduced to one and the same point. When $\boldsymbol{w}_i = \boldsymbol{w}_j$ hold, these two units may be merged into one, and when $v_i + v_j$ is the same, the two points are equivalent even when they differ in $v_i - v_j$. Hence, the dimension reduction takes place in the subspace satisfying $\boldsymbol{w}_i = \boldsymbol{w}_j$. Such singularities occur on the critical submanifolds of the two types

$$1)\ v_i \boldsymbol{w}_i = 0, \qquad 2)\ \boldsymbol{w}_i = \boldsymbol{w}_j. \tag{4}$$

## 3 Simple toy models

Given training data, the parameters of the neural network are estimated or trained by learning. It is important to elucidate the effects of singularities on learning or estimation. We use simple toy models to attack this problem. One is a very simple multilayer perceptron having only one hidden unit. The other is a simple cone model: Let $x$ be Gaussian random variable $x \in R^{d+2}$, with mean $\mu$ and identity covariance matrix $I$,

$$p(x|\mu) = \frac{1}{(\sqrt{2\pi})^{d+2}} \exp\left\{-\frac{1}{2}\|x - \mu\|^2\right\} \tag{5}$$

and let $S = \{\mu|\mu \in R^{d+2}\}$ be the parameter space. The cone model $M$ is a subset of $S$, embedded as

$$M : \mu = \frac{\xi}{\sqrt{1+c^2}} \begin{pmatrix} 1 \\ c\omega \end{pmatrix} = \xi a(\omega) \tag{6}$$

where $c$ is a constant, $\|a^2\| = 1$, $\omega \in S^d$ and $S^d$ is a $d$-dimensional unit sphere. When $d = 1$, $S^1$ is a circle so that $\omega$ is replaced by angle $\theta$, and we have

$$\mu = \frac{\xi}{\sqrt{1+c^2}} \begin{pmatrix} 1 \\ c\cos\theta \\ c\sin\theta \end{pmatrix}. \tag{7}$$

See Figure 1. The $M$ is a cone, having $(\xi, \omega)$ as coordinates, where the apex $\xi = 0$ is the singular point.

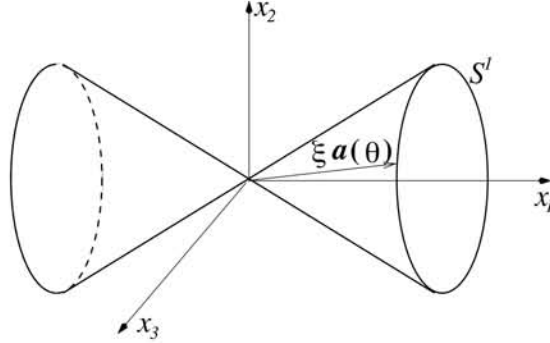

Figure 1: One-dimensional cone model

The input-output relation of a simple multilayer perceptron is given by

$$y = v\varphi(w \cdot x) + n \tag{8}$$

When $v = 0$, the behavior is the same whatever $w$ is. Let us put $w = \beta\omega$, where $\beta = |w|$ and $\omega \in S^d$, and $\xi = v|w|$, $\psi(x; \beta, \omega) = \varphi\{\beta(\omega \cdot x)\}/\beta$. We then have

$$y = \xi\psi(x; \beta, \omega) + n \tag{9}$$

which shows the cone structure with apex at $\xi = 0$. In this paper, we assume that $\beta$ is knwon and does not need to be estimateed.

# 4  Asymptotic statistical inference: generalization error and training error

Let $D = \{\boldsymbol{x}_1, \cdots, \boldsymbol{x}_T\}$ be $T$ independent observations from the true distribution $p_0(\boldsymbol{x})$ which is specified by $\xi = 0$, that is, at the singular point. In the case of neural networks, the training set $D$ is $T$ input-output pairs $(\boldsymbol{x}_t, y_t)$, from the conditional probability distributions $p(y|\boldsymbol{x}; \xi, \boldsymbol{\omega})$ and the true one is at $\xi = 0$. The discussions go in parallel, so that we show here only the cone model. We study the characteristics of both the mle and the Bayesian predictive estimator.

Let $\hat{p}(\boldsymbol{x})$ be the estimated distribution from data $D$. In the case of mle, it is given by $\hat{p}(\boldsymbol{x}; \hat{\boldsymbol{\theta}})$ where $\hat{\boldsymbol{\theta}}$ is the mle given by the maximizer of the log likelihood. For the Bayes estimator, it is given by the Bayes predictive distribution $p(\boldsymbol{x}|D)$.

We evaluate the estimator by the generalization error defined by the KL-divergence from $p_0(\boldsymbol{x})$ to $\hat{p}(\boldsymbol{x})$,

$$E_{gen} = E_D\left[K[p_o : \hat{p}]\right], \quad K[p_o : \hat{p}] = E_{p_o}\left[\log \frac{p_o(\boldsymbol{x})}{\hat{p}(\boldsymbol{x})}\right]. \tag{10}$$

Similarly, the training error is defined by using the empirical expectation,

$$E_{train} = E_D\left[\frac{1}{T}\sum_{t=1}^{T}\log\frac{p_o(\boldsymbol{x}_t)}{\hat{p}(\boldsymbol{x}_t)}\right]. \tag{11}$$

In order to evaluate the estimator $\hat{p}$, one uses $E_{gen}$, but it is not computable. Instead, one uses the $E_{train}$ which is computable. Hence, it is important to see the difference between $E_{gen}$ and $E_{train}$. This is used as a principle of model selection.

When the statistical model $M$ is regular, or the true distribution $p_o(\boldsymbol{x})$ is at a regular point, the mle-based $p(\boldsymbol{x}, \hat{\boldsymbol{\theta}})$ and the Bayes predictive distribution are asymptotically equivalent, and are Fisher efficient under reasonable regularity conditions,

$$E_{gen} \approx \frac{d}{2T}, \quad E_{gen} \approx E_{train} + \frac{d}{T}, \tag{12}$$

where $d$ is the dimension number of parameter vector $\boldsymbol{\theta}$.

All of these good relations do not hold in the singular case. The mle is no more asymptotically Gaussian, the mle and the Bayes estimators have different asymptotic characteristics, although $1/T$ consistency is guaranteed. The relation between the generalization and training error is different, so that we need a different model selection criterion to determine the number of hidden units.

# 5  Gaussian random fields and mle

Here, we introduce the Gaussian random field [11, 12, 13] in the case of the cone model. The log likelihood of data $D$ is written as

$$L(D, \xi, \boldsymbol{\omega}) = -\frac{1}{2}\sum_{t=1}^{T}||\boldsymbol{x}_t - \xi\boldsymbol{a}(\boldsymbol{\omega})||^2. \tag{13}$$

Following Hartigan [13] (see also [11] for details), we first fix $\boldsymbol{\omega}$ and search for the $\xi$ that maximizes $L$. This is easy since $L$ is a quadratic function of $\xi$. The maximum

$\hat{\xi}$ is given by

$$\hat{\xi}(\boldsymbol{\omega}) \quad = \quad \mathrm{argmax}_\xi L(D, \xi, \boldsymbol{\omega}) = \frac{1}{\sqrt{T}} Y(\boldsymbol{\omega}), \tag{14}$$

$$Y(\boldsymbol{\omega}) \quad = \quad \boldsymbol{a}(\boldsymbol{\omega}) \cdot \tilde{\boldsymbol{x}}, \quad \tilde{\boldsymbol{x}} = \frac{1}{\sqrt{T}} \sum_{t=1}^{T} \boldsymbol{x}_t. \tag{15}$$

By the central limit theorem, $Y(\boldsymbol{\omega}) = \boldsymbol{a}(\boldsymbol{\omega}) \cdot \tilde{\boldsymbol{x}}$ is a Gaussian random field defined on $S^d = \{\boldsymbol{\omega}\}$. By substituting $\hat{\xi}(\boldsymbol{w})$ in (14) the log likelihood function becomes

$$\hat{L}(\boldsymbol{\omega}) = -\frac{1}{2} \sum_{t=1}^{T} ||x_t||^2 + \frac{1}{2} Y^2(\boldsymbol{\omega}). \tag{16}$$

Therefore, the mle $\hat{\boldsymbol{w}}$ is given by the maximizer of $\hat{L}(\boldsymbol{w})$, $\hat{\boldsymbol{\omega}} = \mathrm{argmax}_{\boldsymbol{w}} Y^2(\boldsymbol{\omega})$.

**Theorem 1.** In the case of the cone model, the mle satisfies

$$E_{gen} \quad = \quad \frac{1}{2T} E_D \left[ \sup_{\boldsymbol{\omega}} Y^2(\boldsymbol{\omega}) \right], \tag{17}$$

$$E_{train} \quad = \quad -\frac{1}{2T} E_D \left[ \sup_{\boldsymbol{\omega}} Y^2(\boldsymbol{\omega}) \right]. \tag{18}$$

**Corollary 1.** When $d$ is large, the mle satisfies

$$E_{gen} \quad \approx \quad \frac{c^2 d}{2T(1+c^2)}, \tag{19}$$

$$E_{train} \quad \approx \quad -\frac{c^2 d}{2T(1+c^2)}. \tag{20}$$

It should be remarked that the generalization and training errors depend on the shape parameter $c$ as well as the dimension number $d$.

## 6 Bayesian predictive distribution

The Bayes paradigm uses the posterior probability of the parameters based on the set of observations $D$. The posterior probability density is written as,

$$p(\xi, \boldsymbol{\omega}|D) \quad = \quad c(D)\pi(\xi, \boldsymbol{\omega}) \prod_{t=1}^{T} p(\boldsymbol{x}_t|\xi, \boldsymbol{\omega}), \tag{21}$$

where $c(D)$ is the normalization factor depending only on data $D$, $\pi(\xi, \boldsymbol{\omega})$ is a prior distribution on the parameter space. The Bayesian predictive distribution $p(\boldsymbol{x}|D)$ is obtained by averaging $p(\boldsymbol{x}|\xi, \boldsymbol{\omega})$ with respect to the posterior distribution $p(\xi, \boldsymbol{\omega}|D)$, and can be written as

$$p(\boldsymbol{x}|D) = \int p(\boldsymbol{x}|\xi, \boldsymbol{\omega}) p(\xi, \boldsymbol{\omega}|D) d\xi d\boldsymbol{\omega}. \tag{22}$$

The Bayes predictive distribution depends on the prior distribution $\pi(\xi, \boldsymbol{\omega})$. As long as the prior is a smooth function, the first order asymptotic properties are the same for the mle and Bayes estimators in the regular case. However, at singularities, the situation is different. Here, we assume a uniform prior for $\boldsymbol{\omega}$. For $\xi$, we assume two different priors, the uniform prior and the Jeffreys prior.

We show here a sketch of calculations in the case of Jeffreys prior, $\pi(\xi, \boldsymbol{\omega}) \propto |\xi|^d$. By introducing

$$I_d(u) = \frac{1}{\sqrt{2\pi}} \int |z + u|^d \exp\left\{-\frac{1}{2}z^2\right\} dz, \qquad (23)$$

after lengthy calculations, we obtain

$$p(\boldsymbol{x}|D) = \frac{1}{\sqrt{2\pi}^{d+2}} \sqrt{\frac{T}{T+1}}^{-d+1} \exp\left\{-\frac{\|x\|^2}{2}\right\} \frac{P_d(\tilde{\boldsymbol{x}}_{T+1})}{P_d(\tilde{\boldsymbol{x}})}, \qquad (24)$$

where

$$\tilde{\boldsymbol{x}}_{T+1} = \frac{1}{\sqrt{T+1}}(\boldsymbol{x} + \sqrt{T}\tilde{\boldsymbol{x}}), \quad P_d(\tilde{\boldsymbol{x}}) = \int I_d(Y(\boldsymbol{\omega})) \exp\left\{\frac{1}{2}Y^2(\boldsymbol{\omega})\right\} d\boldsymbol{\omega}. \qquad (25)$$

Here $Y(\boldsymbol{\omega})$ has the same form defined in (15), and $P_d(\tilde{\boldsymbol{x}})$ is the function of the sufficient statistics $\tilde{\boldsymbol{x}}$. By using the Edgeworth expansion, we have

$$\begin{aligned} p(\boldsymbol{x}|D) \cong & \frac{1}{\sqrt{2\pi}^{d+2}} \exp\left\{-\frac{\|x\|^2}{2}\right\} \\ & \left\{1 + \frac{1}{\sqrt{T}}\nabla \log P_d(\tilde{\boldsymbol{x}}) \cdot \boldsymbol{x} + \frac{1}{2T} tr\left(\frac{\nabla\nabla P_d}{P_d} H_2(\boldsymbol{x})\right)\right\}, \end{aligned} \qquad (26)$$

where $\nabla$ is the gradient and $H_2(\boldsymbol{x})$ is the Hermite polynomial. We thus have the following theorem.

**Theorem 2.** Under the Jeffreys prior for $\xi$, the generalization error and the training error of the predictive distribution are given by

$$E_{gen} = \frac{1}{2T}E_D\left[\|\nabla \log P_d(\tilde{\boldsymbol{x}})\|^2\right], \qquad (27)$$

$$E_{train} = E_{gen} - \frac{1}{T}E_D\left[\nabla \log P_d(\tilde{\boldsymbol{x}}) \cdot \tilde{\boldsymbol{x}}\right]. \qquad (28)$$

Under the uniform prior, the above results hold by replacing $I_d(Y)$ in the definition of $P_d(\tilde{\boldsymbol{x}})$ by 1. In addition, From (24), we can easily obtain $E_{gen} = (d+1)/2T$ for the Jeffreys prior, and $E_{gen} = 1/2T$ for the uniform prior.

The theorem shows rather surprising results : Under the uniform prior, the generalization error is constant and does not depend on $d$. This is completely different from the regular case. However, this striking result is given rise to by the uniform prior on $\xi$. The uniform prior puts strong emphasis on the singularity, showing that one should be very careful for choosing a prior when the model includes singularities. In the case of Jeffreys prior, the generalization error increases in proportion to $d$, which is the same result as the regular case. In addition, the symmetric duality between $E_{gen}$ and $E_{train}$ does not hold for both of the uniform prior and the Jeffreys prior.

## 7 Gaussian random field of MLP

In the case of MLP with one hidden unit, the log likelihood is written as

$$L(D; \xi, \boldsymbol{\omega}) = -\frac{1}{2}\sum_{t=1}^{T} \{y_t - \xi\varphi_\beta(\boldsymbol{\omega} \cdot \boldsymbol{x}_t)\}^2. \qquad (29)$$

Let us define a Gaussian random field depending on $D$ and $\boldsymbol{\omega}$,

$$Y(\boldsymbol{\omega}) = \frac{1}{\sqrt{T}} \sum_{t=1}^{T} y_t \varphi_\beta (\boldsymbol{\omega} \cdot \boldsymbol{x}_t) \sim N(0, A(\boldsymbol{\omega}, \boldsymbol{\omega}')) \tag{30}$$

where $A(\boldsymbol{\omega}, \boldsymbol{\omega}') = E_{\boldsymbol{x}}[\varphi_\beta(\boldsymbol{\omega} \cdot \boldsymbol{x})\varphi_\beta(\boldsymbol{\omega}' \cdot \boldsymbol{x})]$.

**Theorem 3.**     For the mle, we have

$$\hat{\boldsymbol{\omega}}_{mle} = \mathrm{argmax}_{\boldsymbol{\omega}} Y^2(\boldsymbol{\omega}), \tag{31}$$

$$E_{gen} = \frac{1}{2T} E_D \left[ \sup_{\boldsymbol{\omega}} \frac{Y(\boldsymbol{\omega})^2}{A(\boldsymbol{\omega})} \right], \tag{32}$$

$$E_{train} = -\frac{1}{2T} E_D \left[ \sup_{\boldsymbol{\omega}} \frac{Y(\boldsymbol{\omega})^2}{A(\boldsymbol{\omega})} \right], \tag{33}$$

where $A(\boldsymbol{\omega}) = A(\boldsymbol{\omega}, \boldsymbol{\omega})$.

In order to analyze the Bayes predictive distribution, we define

$$S_d(D, \boldsymbol{\omega}) = \frac{1}{\sqrt{A(\boldsymbol{\omega})}^{d+1}} I_d \left( \frac{Y(\boldsymbol{\omega})}{\sqrt{A(\boldsymbol{\omega})}} \right) \exp \left\{ \frac{1}{2} \frac{Y^2(\boldsymbol{\omega})}{A(\boldsymbol{\omega})} \right\}. \tag{34}$$

We then have the Edgeworth expansion of the predictive distribution of the form,

$$p(y|\boldsymbol{x}, D) \cong \frac{1}{\sqrt{2\pi}} \exp \left\{ -\frac{y^2}{2} \right\} \left\{ 1 + \frac{y}{\sqrt{T}} \frac{E_{\boldsymbol{\omega}}[\nabla S_d(D, \boldsymbol{\omega})\varphi_\beta(\boldsymbol{\omega} \cdot \boldsymbol{x})]}{E_{\boldsymbol{\omega}}[S_d(D, \boldsymbol{\omega})]} \right. \tag{35}$$
$$\left. + \frac{1}{2T} \frac{E_{\boldsymbol{\omega}}[\nabla\nabla S_d(D, \boldsymbol{\omega})A(\boldsymbol{\omega})]}{E_{\boldsymbol{\omega}}[S_d(D, \boldsymbol{\omega})]} H_2(y) \right\},$$

where $\nabla$ is the gradient with respect to $Y(\boldsymbol{\omega})$. We thus have the following theorem.

**Theorem 4.**     Under the Jeffreys prior for $\xi$, the generalization error and the training error of the predictive distribution are given by

$$E_{gen} = \frac{1}{2T} E_D \left[ \frac{E_{\boldsymbol{\omega}\boldsymbol{\omega}'}[\nabla S_d(D, \boldsymbol{\omega})\nabla S_d(D, \boldsymbol{\omega}')A(\boldsymbol{\omega}, \boldsymbol{\omega}')]}{E_{\boldsymbol{\omega}}[S_d(D, \boldsymbol{\omega})]^2} \right],$$

$$E_{train} = E_{gen} - \frac{1}{T} E_D \left[ \frac{E_{\boldsymbol{\omega}}[\nabla S_d(D, \boldsymbol{\omega})Y(\boldsymbol{\omega})]}{E_{\boldsymbol{\omega}}[S_d(D, \boldsymbol{\omega})]} \right]. \tag{36}$$

Under the uniform prior, the above results hold by redefining

$$S_d(D, \boldsymbol{\omega}) = \frac{1}{\sqrt{A}} \exp \left\{ \frac{1}{2} \frac{Y^2(\boldsymbol{\omega})}{A(\boldsymbol{\omega})} \right\}. \tag{37}$$

We can also obtain $E_{gen} = (d+1)/2T$ for the Jeffreys prior, and $E_{gen} = 1/2T$ for the uniform prior.

There is a nice correspondence between the cone model and MLP. However, there is no sufficient statistics in the MLP case, while all the data are summarized in the sufficient statistics $\tilde{\boldsymbol{x}}$ in the cone model.

## 8　Conclusions and discussions

We have analyzed the asymptotic behaviors of the MLE and Bayes estimators in terms of the generalization error and the training error by using simple statistical models (cone model and simple MLP), when the true parameter is at singularity. Since the classic paradigm of statistical inference based on the Cramér-Rao theorem does not hold in such a singular case, we need a new theory. The Gaussian random field has played a fundamental role. We can compare the estimation accuracy of the maximum likelihood estimator and the Bayesian predictive distribution from the results of analysis. Under the proposed framework, the various estimation methods can be studied and compared to each other.

## References

[1] Saad, D. and Solla, S. A. (1995). *Physical Review E*, **52**, 4225-4243.

[2] Amari, S. (1998). *Neural Computation,* **10**, 251-276.

[3] Amari S. and Nagaoka, H. (2000). *Methods of Information Geometry*, AMS.

[4] Amari, S., Park, H., and Fukumizu, F. (2000). *Neural Computation*, **12**, 1399-1409.

[5] Park, H., Amari, S. and Fukumizu, F. (2000). *Neural Networks*, **13**, 755-764.

[6] Chen, A. M., Lu, H., and Hecht-Nielsen, R. (1993). *Neural Computations*, **5**, 910-927.

[7] Rügger, S. M. and Ossen, A. (1997). *Neural Processing Letters*, **5**, 63-72.

[8] Fukumizu, K. and Amari, S. (2000) *Neural Networks*, **13** 317-327.

[9] Hagiwara, K., Hayasaka, K., Toda, N., Usui, S., and Kuno, K. (2001). *Neural Networks*, **14** 1419-1430.

[10] Watanabe, S. (2001). *Neural Computation*, **13**, 899-933.

[11] Fukumizu, K. (2001). *Research Memorandum*, **780**, Inst. of Statistical Mathematics.

[12] Dacunha-Castelle, D. and Gassiat, E. (1997). *Probability and Statistics*, **1**, 285-317.

[13] Hartigan, J. A. (1985). *Proceedings of Berkeley Conference in Honor of J. Neyman and J. Kiefer*, **2**, 807-810.
